# Mortal Multi-Armed Bandits

**Deepayan Chakrabarti**
Yahoo! Research
Sunnyvale, CA 94089
deepay@yahoo-inc.com

**Ravi Kumar**
Yahoo! Research
Sunnyvale, CA 94089
ravikumar@yahoo-inc.com

**Filip Radlinski**[*]
Microsoft Research
Cambridge, UK
filiprad@microsoft.com

**Eli Upfal**[†]
Brown University
Providence, RI 02912
eli@cs.brown.edu

## Abstract

We formulate and study a new variant of the $k$-armed bandit problem, motivated by e-commerce applications. In our model, arms have (stochastic) lifetime after which they expire. In this setting an algorithm needs to continuously explore new arms, in contrast to the standard $k$-armed bandit model in which arms are available indefinitely and exploration is reduced once an optimal arm is identified with near-certainty. The main motivation for our setting is online-advertising, where ads have limited lifetime due to, for example, the nature of their content and their campaign budgets. An algorithm needs to choose among a large collection of ads, more than can be fully explored within the typical ad lifetime.

We present an optimal algorithm for the state-aware (deterministic reward function) case, and build on this technique to obtain an algorithm for the state-oblivious (stochastic reward function) case. Empirical studies on various reward distributions, including one derived from a real-world ad serving application, show that the proposed algorithms significantly outperform the standard multi-armed bandit approaches applied to these settings.

## 1 Introduction

Online advertisements (ads) are a rapidly growing source of income for many Internet content providers. The content providers and the ad brokers who match ads to content are paid only when ads are clicked; this is commonly referred to as the *pay-per-click* model. In this setting, the goal of the ad brokers is to select ads to display from a large corpus, so as to generate the most ad clicks and revenue. The selection problem involves a natural exploration vs. exploitation tradeoff: balancing exploration for ads with better click rates against exploitation of the best ads found so far.

Following [17, 16], we model the ad selection task as a *multi-armed bandit* problem [5]. A multi-armed bandit models a casino with $k$ slot machines (one-armed bandits), where each machine (arm) has a different and unknown expected payoff. The goal is to sequentially select the optimal sequence of slot machines to play (i.e., slot machine arms to pull) to maximize the expected total reward. Considering each ad as a slot machine, that may or may not provide a reward when presented to users, allows any multi-armed bandit strategy to be used for the ad selection problem.

---

[*]Most of this work was done while the author was at Yahoo! Research.

[†]Part of this work was done while the author was visiting the Department of Information Engineering at the University of Padova, Italy, supported in part by the FP6 EC/IST Project 15964 AEOLUS. Work supported in part by NSF award DMI-0600384 and ONR Award N000140610607.

A standard assumption in the multi-armed bandit setting, however, is that each arm exists perpetually. Although the payoff function of an arm is allowed to evolve over time, the evolution is assumed to be slow. Ads, on the other hand, are regularly created while others are removed from circulation. This occurs as advertisers' budgets run out, when advertising campaigns change, when holiday shopping seasons end, and due to other factors beyond the control of the ad selection system. The advertising problem is even more challenging as the set of available ads is often huge (in the tens of millions), while standard multi-armed bandit strategies converge only slowly and require time linear in the number of available options.

In this paper we initiate the study of a rapidly changing variant of the multi-armed bandit problem. We call it the *mortal multi-armed bandit* problem since ads (or equivalently, available bandit arms) are assumed to be born and die regularly. In particular, we will show that while the standard multi-armed bandit setting allows for algorithms that only deviate from the optimal total payoff by $O(\ln t)$ [21], in the mortal arm setting a *regret* of $\Omega(t)$ is possible.

Our analysis of the mortal multi-arm bandit problem considers two settings. First, in the less realistic but simpler *state-aware (deterministic reward)* case, pulling arm $i$ always provides a reward that equals the expected payoff of the arm. Second, in the more realistic *state-oblivious (stochastic reward)* case, the reward from arm $i$ is a binomial random variable indicating the true payoff of the arm only in expectation. We provide an optimal algorithm for the state-aware case. This algorithm is based on characterizing the precise payoff threshold below which repeated arm pulls become suboptimal. This characterization also shows that there are cases when a linear regret is inevitable. We then extend the algorithm to the state-oblivious case, and show that it is near-optimal. Following this, we provide a general heuristic recipe for modifying standard multi-armed bandit algorithms to be more suitable in the mortal-arm setting. We validate the efficacy of our algorithms on various payoff distributions including one empirically derived from real ads. In all cases, we show that the algorithms presented significantly outperform standard multi-armed bandit approaches.

## 2   Modeling mortality

Suppose we wish to select the ads to display on a webpage. Every time a user visits this webpage, we may choose one ad to display. Each ad has a different potential to provide revenue, and we wish to sequentially select the ads to maximize the total expected revenue. Formally, say that at time $t$, we have ads $A(t) = \{ad_{1t}, \ldots, ad_{kt}\}$ from which we must pick one to show. Each $ad_{it}$ has a payoff $\mu_{it} \in [0, 1]$ that is drawn from some known cumulative distribution $F(\mu)$[1]. Presenting $ad_{it}$ at time $t$ provides a (financial) reward $R(\mu_{it})$; the reward function $R(\cdot)$ will be specified below.

If the pool of available ads $A(t)$ were static, or if the payoffs were only slowly changing with $t$, this problem could be solved using any standard multi-armed bandit approach. As described earlier, in reality the available ads are rapidly changing. We propose the following simple model for this change: at the end of each time step $t$, one or more ads may die and be replaced with new ads. The process then continues with time $t + 1$. Note that since change happens only through replacement of ads, the number of ads $k = |A(t)|$ remains fixed. Also, as long as an ad is alive, we assume that its payoff is fixed.

Death can be modeled in two ways, and we will address both in this work. An ad $i$ may have a *budget* $L_i$ that is known a priori and revealed to the algorithm. The ad dies immediately after it has been selected $L_i$ times; we assume that $L_i$ values are drawn from a geometric distribution, with an expected budget of $L$. We refer to this case as *budgeted death*. Alternatively, each ad may die with a fixed probability $p$ after every time step, whether it was selected or not. This is equivalent to each ad being allocated a *lifetime* budget $L_i$, drawn from a geometric distribution with parameter $p$, that is fixed when the arm is born but is never revealed to the algorithm; in this case new arms have an expected lifetime of $L = 1/p$. We call this *timed death*. In both death settings, we assume in our theoretical analysis that at any time there is always at least one previously unexplored ad available. This reflects reality where the number of ads is practically unlimited.

Finally, we model the reward function in two ways, the first being simpler to analyze and the latter more realistic. In the *state-aware (deterministic reward)* case, we assume $R(\mu_{it}) = \mu_{it}$. This

provides us with complete information about each ad immediately after it is chosen to be displayed. In the *state-oblivious (stochastic reward)* case, we take $R(\mu_{it})$ to be a random variable that is 1 with probability $\mu_{it}$ and 0 otherwise.

The mortal multi-armed bandit setting requires different performance measures than the ones used with static multi-armed bandits. In the static setting, very little exploration is needed once an optimal arm is identified with near-certainty; therefore the quality measure is the total regret over time. In our setting the algorithm needs to continuously explore newly available arms. We therefore study the long term, steady-state, mean regret per time step of various solutions. We define this regret as the expected payoff of the best currently alive arm minus the payoff actually obtained by the algorithm.

## 3   Related work

Our work is most related to the study of dynamic versions of the multi-arm bandit (MAB) paradigm where either the set of arms or their expected reward may change over time. Motivated by task scheduling, Gittins [10] proposed a policy where only the state of the active arm (the arm currently being played) can change in a given step, and proved its optimality for the Bayesian formulation with time discounting. This seminal result gave rise to a rich line of work, a proper review of which is beyond the scope of this paper. In particular, Whittle [23] introduced an extension termed *restless bandits* [23, 6, 15], where the states of all arms can change in each step according to a known (but arbitrary) stochastic transition function. Restless bandits have been shown to be intractable: e.g., even with deterministic transitions the problem of computing an (approximately) optimal strategy is PSPACE-hard [18]. *Sleeping bandits* problem, where the set of strategies is fixed but only a subset of them available in each step, were studied in [9, 7] and recently, using a different evaluation criteria, in [13]. Strategies with expected rewards that change gradually over time were studied in [19]. The mixture-of-experts paradigm is related [11], but assumes that data tuples are provided to each expert, instead of the tuples being picked by the algorithm, as in the bandit setting.

Auer et al. [3] adopted an adversarial approach: they defined the *adversarial MAB problem* where the reward distributions are allowed to change arbitrarily over time, and the goal is to approach the performance of the best *time-invariant* policy. This formulation has been further studied in several other papers. Auer et al. [3, 1] also considered a more general definition of regret, where the comparison is to the best policy that can change arms a limited number of times. Due to the overwhelming strength of the adversary, the guarantees obtained in this line of work are relatively weak when applied to the setting that we consider in this paper.

Another aspect of our model is that unexplored arms are always available. Related work broadly comes in three flavors. First, new arms can become available over time; the optimality of Gittins' index was shown to extend to this case [22]. The second case is that of infinite-armed bandits with discrete arms, first studied by [4] and recently extended to the case of unknown payoff distributions and an unknown time horizon [20]. Finally, the bandit arms may be indexed by numbers from the real line, implying uncountably infinite bandit arms, but where "nearby" arms (in terms of distance along the real line) have similar payoffs [12, 14]. However, none of these approaches allows for arms to appear then disappear, which as we show later critically affects any regret bounds.

## 4   Upper bound on mortal reward

In this section we show that in the mortal multi-armed bandit setting, the regret per time step of any algorithm can never go to zero, unlike in the standard MAB setting. Specifically, we develop an upper bound on the mean reward per step of any such algorithm for the state-aware, budgeted death case. We then use reductions between the different models to show that this bound holds for the state-oblivious, timed death cases as well.

We prove the bound assuming we always have new arms available. The expected reward of an arm is drawn from a cumulative distribution $F(\mu)$ with support in $[0, 1]$. For $X \sim F(\mu)$, let $E[X]$ be the expectation of $X$ over $F(\mu)$. We assume that the lifetime of an arm has an exponential distribution with parameter $p$, and denote its expectation by $L = 1/p$. The following function captures the tradeoff between exploration and exploitation in our setting and plays a major role in our analysis:

$$\Gamma(\mu) = \frac{E[X] + (1 - F(\mu))(L - 1)E[X|X \geq \mu]}{1 + (1 - F(\mu))(L - 1)}. \tag{1}$$

**Theorem 1.** *Let $\bar{\mu}(t)$ denote the maximum mean reward that any algorithm for the state-aware mortal multi-armed bandit problem can obtain in $t$ steps in the budgeted death case. Then $\lim_{t\to\infty} \bar{\mu}(t) \leq \max_\mu \Gamma(\mu)$.*

*Proof sketch.* We distinguish between *fresh* arm pulls, i.e., pulls of arms that were not pulled before, and *repeat* arm pulls. Assume that the optimal algorithm pulls $\tau(t)$ distinct (fresh) arms in $t$ steps, and hence makes $t - \tau(t)$ repeat pulls. The expected number of *repeat* pulls to an arm before it expires is $(1-p)/p$. Thus, using Wald's equation [8], the expected number of different arms the algorithm must use for the repeat pulls is $(t - \tau(t)) \cdot p/(1-p)$. Let $\ell(t) \leq \tau(t)$ be the number of distinct arms that get pulled more than once. Using Chernoff bounds, we can show that for any $\delta > 0$, for sufficiently large $t$, with probability $\geq 1 - 1/t^2$ the algorithm uses at least $\ell(t) = p(t - \tau(t))/(1-p) \cdot (1 - \delta)$ different arms for the repeat pulls. Call this event $\mathcal{E}_1(\delta)$.

Next, we upper bound the expected reward of the best $\ell(t)$ arms found in $\tau(t)$ fresh probes. For any $h > 0$, let $\mu(h) = F^{-1}(1 - (\ell(t)/\tau(t))(1 - h))$. In other words, the probability of picking an arm with expected reward greater or equal to $\mu(h)$ is $(\ell(t)/\tau(t))(1 - h)$. Applying the Chernoff bound, for any $\delta, h > 0$ there exists $t(\delta, h)$ such that for all $t \geq t(\delta, h)$ the probability that the algorithm finds at least $\ell(t)$ arms with expected reward at least $\mu(\delta, h) = \mu(h)(1 - \delta)$ is bounded by $1/t^2$. Call this event $\mathcal{E}_2(\delta, h)$.

Let $\mathcal{E}(\delta, h)$ be the event $\mathcal{E}_1(\delta) \wedge \neg\mathcal{E}_2(\delta, h)$. The expected reward of the algorithm in this event after $t$ steps is then bounded by $\tau(t)E[X] + (t - \tau(t))E[X \mid X \geq \mu(\delta, h)]\Pr(\mathcal{E}(\delta, h)) + (t - \tau(t))(1 - \Pr(\mathcal{E}(\delta, h)))$. As $\delta, h \to 0$, $\Pr(\mathcal{E}(\delta, h)) \to 1$, and the expected reward per step when the algorithm pulls $\tau(t)$ fresh arms is given by

$$\limsup_{t\to\infty} \bar{\mu}(t) \leq \frac{1}{t}\Big(\tau(t)E[X] + (t - \tau(t))E[X \mid X \geq \mu]\Big),$$

where $\mu = F^{-1}(1 - \ell(t)/\tau(t))$ and $\ell(t) = (t - \tau(t))p/(1-p)$. After some calculations, we get $\limsup_{t\to\infty} \bar{\mu}(t) \leq \max_\mu \Gamma(\mu)$. $\square$

In Section 5.1 we present an algorithm that achieves this performance bound in the state-aware case.

The following two simple reductions establish the lower bound for the timed death and the state-oblivious models.

**Lemma 2.** *Assuming that new arms are always available, any algorithm for the timed death model obtains at least the same reward per timestep in the budgeted death model.*

Although we omit the proof due to space constraints, the intuition behind this lemma is that an arm in the timed case can die no sooner than in the budgeted case (i.e., when it is always pulled). As a result, we get:

**Lemma 3.** *Let $\bar{\mu}^{det}(t)$ and $\bar{\mu}^{sto}(t)$ denote the respective maximum mean expected rewards that any algorithm for the state-aware and state-oblivious mortal multi-armed bandit problems can obtain after running for t steps. Then $\bar{\mu}^{sto}(t) \leq \bar{\mu}^{det}(t)$.*

We now present two applications of the upper bound. The first simply observes that if the time to find an optimal arm is greater than the lifetime of such an arm, the the mean reward per step of any algorithm must be smaller than the best value. This is in contrast to the standard MAB problem with the same reward distribution, where the mean regret per step tends to 0.

**Corollary 4.** *Assume that the expected reward of a bandit arms is $1$ with probability $p < 1/2$ and $1 - \delta$ otherwise, for some $\delta \in (0, 1]$. Let the lifetime of arms have geometric distribution with the same parameter $p$. The mean reward per step of any algorithm for this supply of arms is at most $1 - \delta + \delta p$, while the maximum expected reward is $1$, yielding an expected regret per step of $\Omega(1)$.*

**Corollary 5.** *Assume arm payoffs are drawn from a uniform distribution, $F(x) = x, x \in [0, 1]$. Consider the timed death case with parameter $p \in (0, 1)$. Then the mean reward per step in bounded by $\frac{1 - \sqrt{p}}{1 - p}$ and expected regret per step of any algorithm is $\Omega(\sqrt{p})$.*

## 5 Bandit algorithms for mortal arms

In this section we present and analyze a number of algorithms specifically designed for the mortal multi-armed bandit task. We develop the optimal algorithm for the state-aware case and then modify

the algorithm to the state-oblivious case, yielding near-optimal regret. We also study a *subset* approach that can be used in tandem with *any* standard multi-armed bandit algorithm to substantially improve performance in the mortal multi-armed bandit setting.

## 5.1 The state-aware case

We now show that the algorithm DETOPT is optimal for this deterministic reward setting.

---
**Algorithm** DETOPT
 **input:** Distribution $F(\mu)$, expected lifetime $L$
 $\mu^* \leftarrow \mathrm{argmax}_\mu \Gamma(\mu)$ &emsp;&emsp;&emsp;&emsp; [$\Gamma$ *is defined in (1)*]
 **while** we keep playing
 &emsp; $i \leftarrow$ random new arm
 &emsp; Pull arm $i$; $R \leftarrow R(\mu_i) = \mu_i$
 &emsp; **if** $R > \mu^*$ &emsp;&emsp;&emsp;&emsp;&emsp; [*If arm is good, stay with it*]
 &emsp;&emsp; Pull arm $i$ every turn until it expires
 &emsp; **end if**
 **end while**

---

Assume the same setting as in the previous section, with a constant supply of new arms. The expected reward of an arm is drawn from cumulative distribution $F(\mu)$. Let $X$ be a random variable with that distribution, and $E[X]$ be its expectation over $F(\mu)$. Assume that the lifetime of an arm has an exponential distribution with parameter $p$, and denote its expectation by $L = 1/p$. Recall $\Gamma(\mu)$ from (1) and let $\mu^* = \mathrm{argmax}_\mu \Gamma(\mu)$. Now,

**Theorem 6.** *Let* DETOPT*(t) denote the mean per turn reward obtained by* DETOPT *after running for t steps with* $\mu^* = \mathrm{argmax}_\mu \Gamma(\mu)$*, then* $\lim_{t\to\infty}$ DETOPT$(t) = \max_\mu \Gamma(\mu)$.

Note that the analysis of the algorithm holds for both budgeted and timed death models.

## 5.2 The state-oblivious case

We now present a modified version of DETOPT for the state-oblivious case. The intuition behind this modification, STOCHASTIC, is simple: instead of pulling an arm once to determine its payoff $\mu_i$, the algorithm pulls each arm $n$ times and abandons it unless it looks promising. A variant, called STOCHASTIC WITH EARLY STOPPING, abandons the arm earlier if its maximum possible future reward will still not justify its retention. For $n = O\left(\log L/\epsilon^2\right)$, STOCHASTIC gets an expected reward per step of $\Gamma(\mu^* - \epsilon)$ and is thus near-optimal; the details are omitted due to space constraints.

---
**Algorithm** STOCHASTIC
 **input:** Distribution $F(\mu)$, expected lifetime $L$
 $\mu^* \leftarrow \mathrm{argmax}_\mu \Gamma(\mu)$ &emsp;&emsp; [$\Gamma$ *is defined in (1)*]
 **while** we keep playing
 &emsp;&emsp;&emsp;&emsp;&emsp; [*Play a random arm n times*]
 &emsp; $i \leftarrow$ random new arm; $r \leftarrow 0$
 &emsp; **for** $d = 1, \ldots, n$
 &emsp;&emsp; Pull arm $i$; $r \leftarrow r + R(\mu_i)$
 &emsp; **end for**
 &emsp; **if** $r > n\mu^*$ &emsp; [*If it is good, stay with it forever*]
 &emsp;&emsp; Pull arm $i$ every turn until it dies
 &emsp; **end if**
 **end while**

---
**Algorithm** STOCH. WITH EARLY STOPPING
 **input:** Distribution $F(\mu)$, expected lifetime $L$
 $\mu^* \leftarrow \mathrm{argmax}_\mu \Gamma(\mu)$ &emsp;&emsp; [$\Gamma$ *is defined in (1)*]
 **while** we keep playing
 &emsp;&emsp;&emsp;&emsp;&emsp; [*Play random arm as long as necessary*]
 &emsp; $i \leftarrow$ random new arm; $r \leftarrow 0$; $d \leftarrow 0$
 &emsp; **while** $d < n$ **and** $n - d \geq n\mu^* - r$
 &emsp;&emsp; Pull arm $i$; $r \leftarrow r + R(\mu_i)$; $d \leftarrow d + 1$
 &emsp; **end while**
 &emsp; **if** $r > n\mu^*$ &emsp; [*If it is good, stay with it forever*]
 &emsp;&emsp; Pull arm $i$ every turn until it dies
 &emsp; **end if**
 **end while**

---

**The subset heuristic.** Why can't we simply use a standard multi-armed bandit (MAB) algorithm for mortal bandits as well? Intuitively, MAB algorithms invest a lot of pulls on all arms (at least logarithmic in the total number of pulls) to guarantee convergence to the optimal arm. This is necessary in the traditional bandit settings, but in the limit as $t \to \infty$, the cost is recouped and leads to sublinear regret. However, such an investment is not justified for mortal bandits: the most gain we can get from an arm is $L$ (if the arm has payoff 1), which reduces the importance of convergence to the best arm. In fact, as shown by Corollary 4, converging to a reasonably good arm suffices.

However, standard MAB algorithms do identify better arms very well. This suggests the following epoch-based heuristic: (a) select a subset of $k/c$ arms uniformly at random from the total $k$ arms at the beginning of each epoch, (b) operate a standard bandit algorithm on these until the epoch ends, and repeat. Intuitively, step (a) reduces the load on the bandit algorithm, allowing it to explore less and converge faster, in return for finding an arm that is probably optimal only among the $k/c$ subset. Picking the right $c$ and the epoch length then depends on balancing the speed of convergence of the bandit algorithm, the arm lifetimes, and the difference between the $k$-th and the $k/c$-th order statistics of the arm payoff distribution; in our experiments, $c$ is chosen empirically.

Using the subset heuristic, we propose an extension of the UCB1 algorithm[2] [2], called UCB1K/C, for the state-oblivious case. Note that this is just one example of the use of this heuristic; any standard bandit algorithm could have been used in place of UCB1 here. In the next section, UCB1K/C is shown to perform far better than UCB1 in the mortal arms setting.

**The** ADAPTIVEGREEDY **heuristic.** Empirically, simple greedy MAB algorithms have previously been shown to perform well due to fast convergence. Hence for the purpose of evaluation, we also compare to an adaptive greedy heuristic for mortal bandits. Note that the $\epsilon_n$-greedy algorithm [2] does not apply directly to mortal bandits since the probability $\epsilon_t$ of random exploration decays to zero for large $t$, which can leave the algorithm with no good choices should the best arm expire.

---

**Algorithm** UCB1K/C
  **input:** $k$-armed bandit, $c$
  **while** we keep playing
    $S \leftarrow k/c$ random arms
    $dead \leftarrow 0$
    $A^{UCB1}(S) \leftarrow$ Initialize UCB1 over arms $S$
    **repeat**
      $i \leftarrow$ arm selected by $A^{UCB1}(S)$
      Pull arm $i$, provide reward to $A^{UCB1}(S)$
      $x \leftarrow$ total arms that died this turn
      Check for newly dead arms in $S$, remove any
      $dead \leftarrow dead + x$
    **until** $dead \geq k/2$ **or** $|S| = 0$
  **end while**

**Algorithm** ADAPTIVEGREEDY
  **input:** $k$-armed bandit, $c$
  Initialization: $\forall i \in [1, k], r_i, n_i \leftarrow 0$
  **while** we keep playing
    $m \leftarrow \operatorname{argmax}_i r_i/n_i$   [*Find best arm so far*]
    $p_m \leftarrow r_m/n_m$
    With probability $\min(1, c \cdot p_m)$
      $j \leftarrow m$
    Otherwise          [*Pull a random arm*]
      $j \leftarrow \operatorname{uniform}(1, k)$
    $r \leftarrow R(j)$
    $r_j \leftarrow r_j + r$   [*Update the observed rewards*]
    $n_j \leftarrow n_j + 1$
  **end while**

---

## 6 Empirical evaluation

In this section we evaluate the performance of UCB1K/C, STOCHASTIC, STOCHASTIC WITH EARLY STOPPING, and ADAPTIVEGREEDY in the mortal arm state-oblivious setting. We also compare these to the UCB1 algorithm [2], that does not consider arm mortality in its policy but is among the faster converging standard multi-armed bandit algorithms. We present the results of simulation studies using three different distributions of arm payoffs $F(\cdot)$.

**Uniform distributed arm payoffs.** Our performance analyses assume that the cumulative payoff distribution $F(\cdot)$ of new arms is known. A particularly simple one is the uniform distribution, $\mu_{it} \sim \operatorname{uniform}(0, 1)$. Figure 1(a) shows the performance of these algorithms as a function of the expected lifetime of each arm, using a timed death and state-oblivious model. The evaluation was performed over $k = 1000$ arms, with each curve showing the mean regret per turn obtained by each algorithm when averaged over ten runs. Each run was simulated for ten times the expected lifetime of the arms, and all parameters were empirically optimized for each algorithm and each lifetime. Repeating the evaluation with $k = 100,000$ arms produces qualitatively very similar performance.

We first note the striking difference between UCB1 and UCB1K/C, with the latter performing far better. In particular, even with the longest lifetimes, each arm can be sampled in expectation at most 100 times. With such limited sampling, UCB1 spends almost all the time exploring and generates almost the same regret of 0.5 per turn as would an algorithm that pulls arms at random.

In contrast, UCB1K/C is able to obtain a substantially lower regret by limiting the exploration to a subset of the arms. This demonstrates the usefulness of the K/C idea: by running the UCB1 algorithm on an appropriately sized *subset* of arms, the overall regret per turn is reduced drastically. In practice,

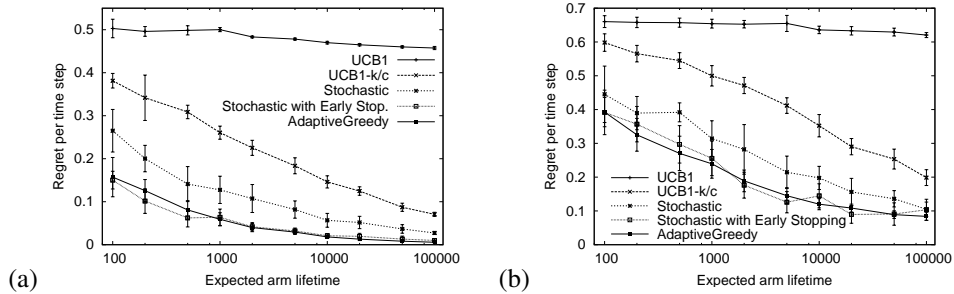

Figure 1: Comparison of the regret per turn obtained by five different algorithms assuming that new arm payoffs come from the (a) uniform distribution and (b) beta$(1, 3)$ distribution.

with $k = 1000$ arms, the best performance was obtained with K/C between 4 and 40, depending on the arm lifetime.

Second, we see that STOCHASTIC outperformed UCB1K/C with optimally chosen parameters. Moreover, STOCHASTIC WITH EARLY STOPPING performs as well as ADAPTIVEGREEDY, which matches the best performance we were able to obtain by any algorithm. This demonstrates that (a) the state-oblivious versions of the optimal deterministic algorithm is effective in general, and (b) the early stopping criterion allows arms with poor payoff to be quickly weeded out.

**Beta distributed arm payoffs.** While the strategies discussed perform well when arm payoffs are uniformly distributed, it is unlikely that in a real setting the payoffs would be so well distributed. In particular, if there are occasional arms with substantially higher payoffs, we could expect any algorithm that does not exhaustively search available arms may obtain very high regret per turn.

Figure 1(b) shows the results when the arm payoff probabilities are drawn from the beta$(1, 3)$ distribution. We chose this distribution as it has finite support yet tends to select small payoffs for most arms while selecting high payoffs occasionally. Once again, we see that STOCHASTIC WITH EARLY STOPPING and ADAPTIVEGREEDY perform best, with the relative ranking of all other algorithms the same as in the uniform case above. The absolute regret of the algorithms we have proposed is increased relative to that seen in Figure 1(a), but still substantially better than that of the UCB1. In fact, the regret of the UCB1 has increased more under this distribution than any other algorithm.

**Real-world arm payoffs.** Considering the application that motivated this work, we now evaluate the performance of the four new algorithms when the arm payoffs come from the empirically observed distribution of clickthrough rates on real ads served by a large ad broker.

Figure 2(a) shows a histogram of the payoff probabilities for a random sample of approximately 300 real ads belonging to a shopping-related category when presented on web pages classified as belonging to the same category. The probabilities have been linearly scaled such that all ads have payoff between 0 and 1. We see that the distribution is unimodal, and is fairly tightly concentrated.

By sampling arm payoffs from a smoothed version of this empirical distribution, we evaluated the performance of the algorithms presented earlier. Figure 2(b) shows that the performance of all the algorithms is consistent with that seen for both the uniform and beta payoff distributions. In particular, while the mean regret per turn is somewhat higher than that seen for the uniform distribution, it is still lower than when payoffs are from the beta distribution. As before, STOCHASTIC WITH EARLY STOPPING and ADAPTIVEGREEDY perform best, indistinguishable from each other.

## 7   Conclusions

We have introduced a new formulation of the multi-armed bandit problem motivated by the real world problem of selecting ads to display on webpages. In this setting the set of strategies available to a multi-armed bandit algorithm changes rapidly over time. We provided a lower bound of linear regret under certain payoff distributions. Further, we presented a number of algorithms that perform substantially better in this setting than previous multi-armed bandit algorithms, including one that is optimal under the state-aware setting, and one that is near-optimal under the state-oblivious setting. Finally, we provided an extension that allows any previous multi-armed bandit algorithm to be used

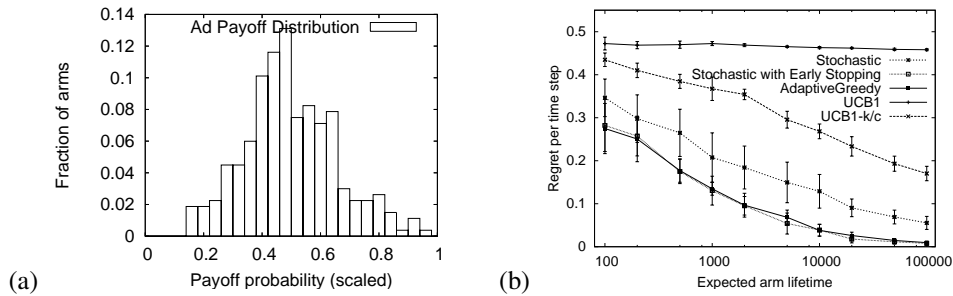

Figure 2: (a) Distribution of real world ad payoffs, scaled linearly such that the maximum payoff is 1 and (b) Regret per turn under the real-world ad payoff distribution.

in the case of mortal arms. Simulations on multiple payoff distributions, including one derived from real-world ad serving application, demonstrate the efficacy of our approach.

### Acknowledgments

We would like to thank the anonymous reviewers for their helpful comments and suggestions.

## Footnotes

[1]We limit our analysis to the case where $F(\mu)$ is stationary and known, as we are particularly interested in the long-term steady-state setting.

[2]UCB1 plays the arm $j$, previously pulled $n_j$ times, with highest mean historical payoff plus $\sqrt{(2 \ln n)/n_j}$.

## References

[1] P. Auer. Using confidence bounds for exploitation-exploration trade-offs. *J. Machine Learning Research*, 3:397–422, 2002.

[2] P. Auer, N. Cesa-Bianchi, and P. Fischer. Finite-time analysis of the multi-armed bandit problem. *Machine Learning*, 47:235–256, 2002.

[3] P. Auer, N. Cesa-Bianchi, Y. Freund, and R. E. Schapire. The nonstochastic multiarmed bandit problem. *SIAM J. Comput.*, 32(1):48–77, 2002.

[4] D. A. Berry, R. W. Chen, A. Zame, D. C. Heath, and L. A. Shepp. Bandit problems with infinitely many arms. *The Annals of Statistics*, 25(5):2103–2116, 1997.

[5] D. A. Berry and B. Fristedt. *Bandit Problems: Sequential Allocation of Experiments*. Chapman and Hall, London, UK, 1985.

[6] D. Bertsimas and J. Nino-Mora. Restless bandits, linear programming relaxations, and a primal-dual index heuristic. *Operations Research*, 48(1):80–90, 2000.

[7] A. Blum and Y. Mansour. From external to internal regret. In *18th COLT*, pages 621–636, 2005.

[8] W. Feller. *An Introduction to Probability Theory and Its Applications, Volume 2*. Wiley, 1971.

[9] Y. Freund, R. Schapire, Y. Singer, and M. K. Warmuth. Using and combining predictors that specialize. In *29th STOC*, pages 334–343, 1997.

[10] J. C. Gittins and D. M. Jones. A dynamic allocation index for the sequential design of experiments. In J. G. et al., editor, *Progress in Statistics*, pages 241–266. North-Holland, 1974.

[11] M. Herbster and M. K. Warmuth. Tracking the best expert. *Machine Learning*, 32:151–178, 1998.

[12] R. Kleinberg. *Online Decision Problems with Large Strategy Sets*. PhD thesis, MIT, 2005.

[13] R. D. Kleinberg, A. Niculescu-Mizil, and Y. Sharma. Regret bounds for sleeping experts and bandits. In *21st COLT*, pages 425–436, 2008.

[14] A. Krause and C. Guestrin. Nonmyopic active learning of Gaussian processes: An exploration-exploitation approach. In *24th ICML*, pages 449–456, 2007.

[15] J. Nino-Mora. Restless bandits, partial conservation laws and indexability. *Adv. Appl. Prob.*, 33:76–98, 2001.

[16] S. Pandey, D. Agarwal, D. Chakrabarti, and V. Josifovski. Bandits for taxonomies: A model-based approach. In *SDM*, pages 216–227, 2007.

[17] S. Pandey, D. Chakrabarti, and D. Agarwal. Multi-armed bandit problems with dependent arms. In *ICML*, pages 721–728, 2007.

[18] C. H. Papadimitriou and J. N. Tsitsiklis. The complexity of optimal queueing network control. In *9th CCC*, pages 318–322, 1994.

[19] A. Slivkins and E. Upfal. Adapting to a changing environment: The Brownian restless bandits. In *21st COLT*, pages 343–354, 2008.

[20] O. Teytaud, S. Gelly, and M. Sebag. Anytime many-armed bandits. In *CAP*, 2007.

[21] T.Lai and H.Robbins. Asymptotically efficient adaptive allocation rules. *Adv. Appl. Math.*, 6:4–22, 1985.

[22] P. Whittle. Arm-acquiring bandits. *The Annals of Probability*, 9(2):284–292, 1981.

[23] P. Whittle. Restless bandits: Activity allocation in a changing world. *J. of Appl. Prob.*, 25A:287–298, 1988.
